# Generalization Error Bounds for Collaborative Prediction with Low-Rank Matrices

**Nathan Srebro**
Department of Computer Science
University of Toronto
Toronto, ON, Canada
nati@cs.toronto.edu

**Noga Alon**
School of Mathematical Sciences
Tel Aviv University
Ramat Aviv, Israel
nogaa@tau.ac.il

**Tommi S. Jaakkola**
Computer Science and Artificial Intelligence Laboratory
Massachusetts Institute of Technology
Cambridge, MA, USA
tommi@csail.mit.edu

## Abstract

We prove generalization error bounds for predicting entries in a partially observed matrix by fitting the observed entries with a low-rank matrix. In justifying the analysis approach we take to obtain the bounds, we present an example of a class of functions of finite pseudodimension such that the sums of functions from this class have unbounded pseudodimension.

## 1 Introduction

"Collaborative filtering" refers to the general task of providing users with information on what items they might like, or dislike, based on their preferences so far and how they relate to the preferences of other users. This approach contrasts with a more traditional feature-based approach where predictions are made based on features of the items.

For feature-based approaches, we are accustomed to studying prediction methods in terms of probabilistic post-hoc generalization error bounds. Such results provide us a (probabilistic) bound on the performance of our predictor on future examples, in terms of its performance on the training data. These bounds hold without any assumptions on the true "model", that is the true dependence of the labels on the features, other than the central assumptions that the training examples are drawn i.i.d. from the distribution of interest.

In this paper we suggest studying the generalization ability of collaborative prediction methods. By "collaborative prediction" we indicate that the objective is to be able to predict user preferences for items, that is, entries in some unknown *target matrix $Y$* of user-item "ratings", based on observing a subset $Y_S$ of the entries in this matrix[1]. We present

| | | |
|---|:---:|---|
| arbitrary source distribution | ⇔ | target matrix $Y$ |
| random training set | ⇔ | random set $S$ of observed entries |
| hypothesis | ⇔ | predicted matrix $X$ |
| training error | ⇔ | observed discrepancy $\mathcal{D}_S(X; Y)$ |
| generalization error | ⇔ | true discrepancy $\mathcal{D}(X; Y)$ |

Figure 1: Correspondence with post-hoc bounds on the generalization error for standard feature-based prediction tasks

bounds on the true average overall error $\mathcal{D}(X; Y) = \frac{1}{nm} \sum_{i=1}^{n} \sum_{a=1}^{m} \text{loss}(X_{ia}; Y_{ia})$ of the predictions $X$ in terms of the average error over the observed entries $\mathcal{D}_S(X; Y) = \frac{1}{|S|} \sum_{ia \in S} \text{loss}(X_{ia}; Y_{ia})$, without making any assumptions on the true nature of the preferences $Y$. What we do assume is that the subset $S$ of entries that we observe is chosen uniformly at random. This strong assumption parallels the i.i.d. source assumption for feature-based prediction.

In particular, we present generalization error bounds on prediction using low-rank models.

Collaborative prediction using low-rank models is fairly straight forward. A low-rank matrix $X$ is sought that minimizes the average observed error $\mathcal{D}_S(X; Y)$. Unobserved entries in $Y$ are then predicted according to $X$. The premise behind such a model is that there are only a small number of factors influencing the preferences, and that a user's preference vector is determined by how each factor applies to that user. Different methods differ in how they relate real-valued entries in $X$ to preferences in $Y$, and in the associated measure of discrepancy. For example, entries in $X$ can be seen as parameters for a probabilistic models of the entries in $Y$, either mean parameters [1] or natural parameters [2], and a maximum likelihood criterion used. Or, other loss functions, such as squared error [3, 2], or zero-one loss versus the signs of entries in $X$, can be minimized.

**Prior Work** Previous results bounding the error of collaborative prediction using a low-rank matrix all assume the true target matrix $Y$ is well-approximated by a low-rank matrix. This corresponds to a large *eigengap* between the top few singular values of $Y$ and the remaining singular values. Azar *et al* [3] give asymptotic results on the convergence of the predictions to the true preferences, assuming they have an eigengap. Drineas *et al* [4] analyze the sample complexity needed to be able to predict a matrix with an eigengap, and suggests strategies for actively querying entries in the target matrix. To our knowledge, this is the first analysis of the generalization error of low-rank methods that do not make any assumptions on the true target matrix.

Generalization error bounds (and related online learning bounds) were previously discussed for collaborative prediction applications, but only when prediction was done for each user separately, using a feature-based method, with the other user's preferences as features [5, 6]. Although these address a collaborative prediction application, the learning setting is a standard feature-based setting. These methods are also limited, in that learning must be performed separately for each user.

Shaw-Taylor *et al* [7] discuss assumption-free post-hoc bounds on the residual errors of low-rank approximation. These results apply to a different setting, where a subset of the rows are fully observed, and bound a different quantity—the distance between rows and the learned *subspace*, rather then the distance to predicted entries.

**Organization** In Section 2 we present a generalization error bound for zero-one loss, based on a combinatorial result which we prove in Section 3. In Section 4 we generalize the bound to arbitrary loss functions. Finally, in Section 5 we justify the combinatorial

approach taken, by considering an alternate approach (viewing rank-$k$ matrices as combination of $k$ rank-1 matrices) and showing why it does not work.

## 2 Generalization Error Bound for Zero-One Error

We begin by considering binary labels $Y_{ia} \in \pm$ and a zero-one sign agreement loss:

$$\text{loss}^{\pm}(X_{ia}; Y_{ia}) = \mathbf{1}_{Y_{ia} X_{ia} \leq 0} \tag{1}$$

**Theorem 1.** *For any matrix $Y \in \{\pm 1\}^{n \times m}$, $n, m > 2$, $\delta > 0$ and integer $k$, with probability at least $1 - \delta$ over choosing a subset $S$ of entries in $Y$ uniformly among all subsets of $|S|$ entries, the discrepancy with respect to the zero-one sign agreement loss satisfies[2]:*

$$\forall_{X,\text{rank } X < k} \mathcal{D}^{\pm}(X; Y) < \mathcal{D}_S^{\pm}(X; Y) + \sqrt{\frac{k(n+m)\log \frac{16em}{k} - \log \delta}{2|S|}}$$

To prove the theorem we employ standard arguments about the generalization error for finite hypothesis classes with bounded cardinality.

First fix $Y$ as well as $X \in \mathbb{R}^{n \times m}$. When an index pair $(i, a)$ is chosen uniformly at random, $\text{loss}(X_{ia}; Y_{ia})$ is a Bernoulli random variable with probability $\mathcal{D}^{\pm}(X; Y)$ of being one. If the entries of $S$ are chosen independently and uniformly, $|S|\mathcal{D}_S^{\pm}(X; Y)$ is Binomially distributed with mean $|S|\mathcal{D}^{\pm}(X; Y)$ and using Chernoff's inequality:

$$\Pr_S \left( \mathcal{D}^{\pm}(X; Y) \geq \mathcal{D}_S^{\pm}(X; Y) + \epsilon \right) \leq e^{-2|S|\epsilon^2} \tag{2}$$

The distribution of $S$ in Theorem 1 is slightly different, as $S$ is chosen without repetitions. The mean of $\mathcal{D}_S^{\pm}(X; Y)$ is the same, but it is more concentrated, and (2) still holds.

Now consider all rank-$k$ matrices. Noting that $\text{loss}(X_{ia}; Y_{ia})$ depends only on the *sign* of $X_{ia}$, it is enough to consider the equivalence classes of matrices with the same sign patterns. Let $f(n, m, k)$ be the number of such equivalence classes, i.e. the number of possible sign configurations of $n \times m$ matrices of rank at most $k$:

$$F(n, m, k) = \{\text{sign } X \in \{-, 0, +\}^{n \times m} | X \in \mathbb{R}^{n \times m}, \text{rank } X \leq k\}$$
$$f(n, m, k) = \sharp F(n, m, k)$$

where sign $X$ denotes the element-wise sign matrix $(\text{sign } X)_{ia} = \begin{cases} 1 & \text{If } X_{ia} > 0 \\ 0 & \text{If } X_{ia} = 0 \\ -1 & \text{If } X_{ia} < 1 \end{cases}$.

For all matrices in an equivalence class, the random variable $\mathcal{D}_S^{\pm}(X; Y)$ is the same, and taking a union bound of the events $\mathcal{D}^{\pm}(X; Y) \geq \mathcal{D}_S^{\pm}(X; Y) + \epsilon$ for each of these $f(n, m, k)$ random variables we have:

$$\Pr_S \left( \exists_{X,\text{rank } X \leq k} \mathcal{D}^{\pm}(X; Y) \geq \mathcal{D}_S^{\pm}(X; Y) + \sqrt{\frac{\log f(n, m, k) - \log \delta}{2|S|}} \right) \leq \delta \tag{3}$$

by using (2) and setting $\epsilon = \sqrt{\frac{\log f(n,m,k) - \log \delta}{2|S|}}$. The proof of Theorem 1 rests on bounding $f(n, m, k)$, which we will do in the next section.

Note that since the equivalence classes we defined do not depend on the sample set, no symmetrization argument is necessary.

## 3 Sign Configurations of a Low-Rank Matrix

In this section, we bound the number $f(n, m, k)$ of sign configurations of $n \times m$ rank-$k$ matrices over the reals. Such a bound was previously considered in the context of unbounded error communication complexity. Alon, Frankl and Rödl [8] showed that $f(n, m, k) \leq \min_h (8\lceil nm/h \rceil)^{(n+m)k+h+m}$, and used counting arguments to establish that some (in fact, most) binary matrices can only be realized by high-rank matrices, and therefore correspond to functions with high unbounded error communication complexity.

Here, we follow a general course outlined by Alon [9] to obtain a simpler, and slightly tighter, bound based on the following result due to Warren:

Let $P_1, \ldots, P_r$ be real polynomials in $q$ variables, and let $C$ be the complement of the variety defined by $\Pi_i P_i$, i.e. the set of points in which all the $m$ polynomials are non-zero:

$$C = \{x \in \mathbb{R}^q | \forall_i P_i(x) \neq 0\}$$

**Theorem 2 (Warren [10]).** *If all $r$ polynomials are of degree at most $d$, then the number of connected components of $C$ is at most:*

$$c(C) \leq 2(2d)^q \sum_{i=0}^{q} 2^i \binom{r}{i} \leq \left( \frac{4edr}{q} \right)^q$$

*where the second inequality holds when $r > q > 2$.*

The signs of the polynomials $P_1, \ldots, P_r$ are fixed inside each connected component of $C$. And so, $c(C)$ bounds the number of sign configurations of $P_1, \ldots, P_r$ that *do not contain zeros*. To bound the overall number of sign configurations the polynomials are modified slightly (see Appendix), yielding:

**Corollary 3 ([9, Proposition 5.5]).** *The number of -/0/+ sign configurations of $r$ polynomials, each of degree at most $d$, over $q$ variables, is at most $(8edr/q)^q$ (for $r > q > 2$).*

In order to apply these bounds to low-rank matrices, recall that any matrix $X$ of rank at most $k$ can be written as a product $X = UV'$ where $U \in \mathbb{R}^{n \times k}$ and $V \in \mathbb{R}^{k \times m}$. Consider the $k(n+m)$ entries of $U, V$ as variables, and the $nm$ entries of $X$ as polynomials of degree two over these variables:

$$X_{ia} = \sum_{\alpha=1}^{k} U_{i\alpha} V_{a\alpha}$$

Applying Corollary 3 we obtain:

**Lemma 4.** $f(n, m, k) \leq \left( \frac{8e \cdot 2 \cdot nm}{k(n+m)} \right)^{k(n+m)} \leq (16em/k)^{k(n+m)}$

Substituting this bound in (3) establishes Theorem 1. The upper bound on $f(n, m, k)$ is tight up to a multiplicative factor in the exponent:

**Lemma 5.** *For $m > k^2$, $f(n, m, k) \geq m^{\frac{1}{2}(k-1)n}$*

*Proof.* Fix any matrix $V \in \mathbb{R}^{m \times k}$ with rows in general position, and consider the number $f(n, V, k)$ of sign configurations of matrices $UV'$, where $U$ varies over all $n \times k$ matrices. Focusing only on $+/-$ sign configurations (no zeros in $UV'$), each row of sign $UV'$ is a homogeneous linear classification of the rows of $V$, i.e. of $m$ vectors in general position in $\mathbb{R}^k$. There are exactly $\left( 2 \sum_{i=0}^{k-1} \binom{m}{i} \right)$ possible homogeneous linear classifications of $m$ vectors in general position in $\mathbb{R}^k$, and so these many options for each row of sign $UV'$. We can therefore bound:

$$f(n, m, k) \geq f(n, V, k) \geq \left( 2 \sum_{i=0}^{k-1} \binom{m}{i} \right)^n \geq \binom{m}{k-1}^n \geq \left( \frac{m}{k-1} \right)^{n(k-1)} = m^{\frac{1}{2}(k-1)n} \quad \square$$

# 4 Generalization Error Bounds for Other Loss Functions

In Section 2 we considered generalization error bounds for a zero-one loss function. More commonly, though, other loss functions are used, and it is desirable to obtain generalization error bounds for general loss functions.

When dealing with other loss functions, the magnitude of the entries in the matrix are important, and not only their signs. It is therefore no longer enough to bound the number of sign configurations. Instead, we will bound not only the number of ways low rank matrices behave with regards to a threshold of zero, but the number of possible ways low-rank matrices can behave relative to any set of thresholds. That is, for any threshold matrix $T \in \mathbb{R}^{n \times m}$, we will show that the number of possible sign configurations of $(X - T)$, where $X$ is low-rank, is small. Intuitively, this captures the complexity of the class of low-rank matrices not only around zero, but throughout all possible values.

We then use standard results from statistical machine learning to obtain generalization error bounds from the bound on the number of relative sign configurations. The number of relative sign configurations serves as a bound on the *pseudodimension*—the maximum number of entries for which there exists a set of thresholds such that all relative sign configurations (limited to these entries) is possible. The pseudodimension can in turn be used to show the existence of a small $\epsilon$-net, which is used to obtain generalization error bounds.

Recall the definition of the pseudodimension of a class of real-valued functions:

**Definition 1.** *A class $\mathcal{F}$ of real-valued functions* pseudo-shatters *the points $x_1, \ldots, x_n$ with thresholds $t_1, \ldots, t_n$ if for every binary labeling of the points $(s_1, \ldots, s_n) \in \{+, -\}^n$ there exists $f \in \mathcal{F}$ s.t. $f(x_i) \leq t_i$ iff $s_i = -$. The* pseudodimension *of a class $\mathcal{F}$ is the supremum over $n$ for which there exist $n$ points and thresholds that can be shattered.*

In order to apply known results linking the pseudodimension to covering numbers, we consider matrices $X \in \mathbb{R}^{n \times m}$ as real-valued functions $X : [n] \times [m] \to \mathbb{R}$ over index pairs to entries in the matrix. The class $\mathcal{X}_k$ of rank-$k$ matrices can now be seen as a class of real-valued functions over the domain $[n] \times [m]$. We bound the pseudodimension of this class by bounding, for any threshold matrix $T \in \mathbb{R}^{n \times m}$ the number of *relative sign matrices*:

$$F_T(n, m, k) = \{\text{sign}\, (X - T) \in \{-, 0, +\}^{n \times m} | X \in \mathbb{R}^{n \times m}, \text{rank}\, X \leq k\}$$
$$f_T(n, m, k) = \sharp F_T(n, m, k)$$

**Lemma 6.** *For any $T \in \mathbb{R}^{n \times m}$, we have $f_T(n, m, k) \leq \left(\frac{16em}{k}\right)^{k(n+m)}$.*

*Proof.* We take a similar approach to that of Lemma 4, writing rank-$k$ matrices as a product $X = UV'$ where $U \in \mathbb{R}^{n \times k}$ and $V \in \mathbb{R}^{k \times m}$. Consider the $k(n + m)$ entries of $U, V$ as variables, and the $nm$ entries of $X - T$ as polynomials of degree two over these variables:

$$(X - T)_{ia} = \sum_{\alpha=1}^{k} U_{i\alpha} V_{a\alpha} - T_{ia}$$

Applying Corollary 10 yields the desired bound. $\square$

**Corollary 7.** *The pseudodimension of the class $\mathcal{X}_k$ of $n \times m$ matrices over the reals of rank at most $k$, is at most $k(n + m) \log \frac{16em}{k}$.*

We can now invoke standard generalization error bounds in terms of the pseudodimension (Theorem 11 in the Appendix) to obtain:

**Theorem 8.** *For any monotone loss function with $|loss| \leq M$, any matrix $Y \in \{\pm 1\}^{n \times m}$, $n, m > 2$, $\delta > 0$ and integer $k$, with probability at least $1 - \delta$ over choosing a subset $S$ of entries in $Y$ uniformly among all subsets of $|S|$ entries:*

$$\forall_{X, \text{rank } X < k} \mathcal{D}(X; Y) < \mathcal{D}_S(X; Y) + 6\sqrt{\frac{k(n+m) \log \frac{16em}{k} \log \frac{M|S|}{k(n+m)} - \log \delta}{|S|}}$$

## 5   Low-Rank Matrices as Combined Classifiers

Rank-$k$ matrices are those matrices which are a sum of $k$ rank-1 matrices. If we view matrices as functions from pairs of indices to the reals, we can think of rank-$k$ matrices as "combined" classifiers, and attempt to bound their complexity as such, based on the low complexity of the "basis" functions, i.e. rank-1 matrices.

A similar approach is taken in related work on learning with low-norm (maximum margin) matrix factorization [11, 12], where the hypothesis class can be viewed as a convex combination of rank-1 unit-norm matrices. Scale-sensitive (i.e. dependent on the margin, or the slope of the loss function) generalization error bounds for this class are developed based on the graceful behavior of scale-sensitive complexity measures (e.g. log covering numbers and the Rademacher complexity) with respect to convex combinations. Taking a similar view, it is possible to obtain scale-sensitive generalization error bounds for low-rank matrices. In this Section we question whether it is possible to obtain scale-insensitive bounds, similar to Theorems 1 and 8, by viewing low-rank matrices as combined classifiers.

It cannot be expected that scale-insensitive complexity would be preserved when taking convex combinations of an unbounded number of base functions. However, the VC-dimension, a scale-insensitive measure of complexity, does scale gracefully when taking linear combinations of a bounded number of functions from a low VC-dimension class of *indicator function*. Using this, we can obtain generalization error bounds for linear combinations of signs of rank-one matrices, but not signs of linear combinations of rank-one matrices. An alternate candidate scale-insensitive complexity measure is the pseudodimension of a class of real-valued functions. If we could bound the pseudodimension of the class of sums of $k$ functions from a bounded-pseudodimension base class of real valued functions, we could avoid the sign-configuration counting and obtain generalization error bounds for rank-$k$ matrices. Unfortunately, the following counterexample shows that this is not possible.

**Theorem 9.** *There exists a family $\mathcal{F}$ closed under scalar multiplication whose pseudodimension is at most five, and such that $\{f_1 + f_2 | f_1, f_2 \in \mathcal{F}\}$ does not have a finite pseudodimension.*

*Proof.* We describe a class $\mathcal{F}$ of real-valued functions over the positive integers $\mathbb{N}$. To do so, consider a one-to-one mapping of *finite* sets of positive integers to the positive integers. For each $A \in \mathbb{N}$ define two functions[3], $f_A(x) = 2^{xA} + \mathbf{1}_{x \in A}$ and $g_A(x) = 2^{xA}$. Let $\mathcal{F}$ be the set of all scalar multiplications of these functions.

For every $A \subset N$, $f_A - g_A$ is the indicator function of $A$, implying that every finite subset can be shattered, and the pseudodimension of $\{f_1 + f_2 : f_1, f_2 \in \mathcal{F}\}$ is unbounded.

It remains to show that the pseudodimension of $\mathcal{F}$ is less than six. To do so, we note that there are no positive integers $A < B$ and $x < y$ and positive reals $\alpha, \beta > 0$ such that $\beta(2^{xB} + 1) > \alpha 2^{xA}$ and $\beta 2^{yB} < \alpha(2^{yA} + 1)$. It follows that for any $A < B$ and any $\alpha, \beta > 0$, on an initial segment (possibly empty) of $\mathbb{N}$ we have $\beta g_B \leq \beta f_B \leq \alpha g_A \leq \alpha f_A$ while on the rest of $\mathbb{N}$ we have $\alpha g_A \leq \alpha f_A < \beta g_B \leq \beta f_B$. In particular, any pair of

functions $(\beta f_A, \alpha f_B)$ or $(\beta f_A, \alpha g_B)$ or $(\beta g_A, \alpha g_B)$ in $\mathcal{F}$ that are not associated with the same subset (i.e. $A \neq B$), cross each other at most once. This holds also when $\alpha$ or $\beta$ are negative, as the functions never change signs.

For any six naturals $x_1 < x_2 < \cdots < x_6$ and six thresholds, consider the three labellings $(+,-,+,-,+,-),(-,+,-,+,-,+),(+,+,-,-,+,+)$. The three functions realizing these labellings must cross each other at least twice, but by the above arguments, there are no three functions in $\mathcal{F}$ such that every pair crosses each other at least twice.[4]   $\square$

## 6   Discussion

Alon, Frankl and Rödl [8] use a result of Milnor similar to Warren's Theorem 2. Milnor's and Warren's theorems were previously used for bounding the VC-dimension of certain geometric classes [13], and of general concept classes parametrized by real numbers, in terms of the complexity of the boolean formulas over polynomials used to represent them [14]. This last general result can be used to bound the VC-dimension of signs of $n \times m$ rank-$k$ matrices by $2k(n+m)\log(48enm)$, yielding a bound similar to Theorem 1 with an extra $\log|S|$ term. In this paper, we take a simpler path, applying Warren's theorem directly, and thus avoiding the $\log|S|$ term and reducing the other logarithmic term. Applying Warren's theorem directly also enables us to bound the pseudodimension and obtain the bound of Theorem 8 for general loss functions.

Another notable application of Milnor's result, which likely inspired these later uses, is for bounding the number of configurations of $n$ points in $\mathbb{R}^d$ with different possible linear classifications [15, 16]. Viewing signs of rank-$k$ $n \times m$ matrices as $n$ linear classification of $m$ points in $\mathbb{R}^k$, this bound can be used to bound $f(n,m,k) < 2^{km \log 2n + k(k+1)n \log n}$ without using Warren's Theorem directly [8, 12]. The bound of Lemma 4 avoids the quadratic dependence on $k$ in the exponent.

**Acknowledgments**   We would like to thank Peter Bartlett for pointing out [13, 14]. N.S. and T.J. would like to thank Erik Demaine for introducing them to oriented matroids.

## A   Proof of Corollary 3

Consider a set $R \subset \mathbb{R}^q$ containing one variable configuration for each possible sign pattern. Set $\epsilon \doteq \frac{1}{2} \min_{1 \leq i \leq q, x \in R P_i(x) \neq 0} |P_i(x)| > 0$. Now consider the $2q$ polynomials $P_i^+(x) = P_i(x) + \epsilon$ and $P_i^-(x) = P_i(x) - \epsilon$ and $C' = \{x \in \mathbb{R}^q | \forall_i P_i^+(x) \neq 0, P_i^-(x) \neq 0\}$. Different points in $R$ (representing all sign configurations) lie in different connected components of $C'$. Invoking Theorem 2 on $C'$ establishes Corollary 3.

The count in Corollary 3 differentiates between positive, negative and zero signs. However, we are only concerned with the positivity of $Y_{ia} X_{ia}$ (in the proof of Theorem 1) or of $X_{ia} - T_{ia}$ (in the proof of Theorem 8), and do not need to differentiate between zero and negative values. Invoking Theorem 2 on $C^+ = \{x \in \mathbb{R}^q | \forall_i P_i^+(x) \neq 0\}$, yields:

**Corollary 10.** *The number of -/+ sign configurations (where zero is considered negative) of $r$ polynomials, each of degree at most $d$, over $q$ variables, is at most $(4edr/q)^q$ (for $r > q > 2$).*

Applying Corollary 10 on the $nm$ degree-two polynomials $Y_{ia} \sum_{\alpha=1}^{k} U_{i\alpha} V_{a\alpha}$ establishes that for any $Y$, the number of configurations of sign agreements of rank-$k$ matrices with $Y$ is bounded by $(8em/k)^{k(n+m)}$ and yields a constant of 8 instead of 16 inside the logarithm in Theorem 1. Applying Corollary 10 instead of Corollary 3 allows us to similarly tighten in the bounds in Corollary 7 and in Theorem 8.

# B  Generalization Error Bound in terms of the Pseudodimension

**Theorem 11.** *Let $\mathcal{F}$ be a class of real-valued functions $f : \mathcal{X} \to \mathbb{R}$ with pseudodimension d, and loss $: \mathbb{R} \times \mathcal{Y} \to \mathbb{R}$ be a bounded monotone loss function (i.e. for all $y$, $loss(x, y)$ is monotone in $x$), with loss $< M$. For any joint distribution over $(X, Y)$, consider an i.i.d. sample $S = (X_1, Y_1), \ldots, (X_n, Y_n)$. Then for any $\epsilon > 0$:*

$$\Pr_S \left( \exists_{f \in \mathcal{F}} \mathbf{E}_{X,Y} \left[ loss(f(X), Y) \right] > \frac{1}{n} \sum_{i=1}^{n} loss(f(X_i), Y_i) + \epsilon \right) < 4e(d+1) \left( \frac{32eM}{\epsilon} \right)^d e^{-\frac{\epsilon^2 n}{32}}$$

The bound is a composition of a generalization error bound in terms of the $L_1$ covering number [17, Theorem 17.1], a bound on the $L_1$ covering number in terms of the pseudodimension [18] and the observation that composition with a monotone function does not increase the pseudodimension [17, Theorem 12.3].

## Footnotes

[1]In other collaborative filtering tasks, the objective is to be able to provide each user with a few items that overlap his top-rated items, while it is not important to be able to correctly predict the users ratings for other items. Note that it is possible to derive generalization error bounds for this objective based on bounds for the "prediction" objective.

[2] All logarithms are base two

[3]We use $A$ to refer both to a positive integer and the finite set it maps to.

[4] A more careful analysis shows that $\mathcal{F}$ has pseudodimension three.

# References

[1] T. Hofmann. Latent semantic models for collaborative filtering. *ACM Trans. Inf. Syst.*, 22(1):89–115, 2004.

[2] Nathan Srebro and Tommi Jaakkola. Weighted low rank approximation. In *20th International Conference on Machine Learning*, 2003.

[3] Yossi Azar, Amos Fiat, Anna R. Karlin, Frank McSherry, and Jared Saia. Spectral analysis of data. In *ACM Symposium on Theory of Computing*, pages 619–626, 2001.

[4] Petros Drineas, Iordanis Kerenidis, and Prabhakar Raghavan. Competitive recommendation systems. In *ACM Symposium on Theory of Computing*, 2002.

[5] K. Crammer and Y. Singer. Pranking with ranking. In *Advances in Neural Information Processing Systems*, volume 14, 2002.

[6] Sanjoy Dasgupta, Wee Sun Lee, and Philip M. Long. A theoretical analysis of query selection for collaborative filtering. *Machine Learning*, 51(3):283–298, 2003.

[7] John Shawe-Taylor, Nello Cristianini, and Jaz Kandola. On the concentration of spectral properties. In *Advances in Neural Information Processing Systems*, volume 14, 2002.

[8] N. Alon, P Frankl, and V. Rödel. Geometrical realization of set systems and probabilistic communication complexity. In *Foundations of Computer Science (FOCS)*, 1985.

[9] Noga Alon. Tools from higher algebra. In M. Grötschel R.L. Graham and L. Lovász, editors, *Handbook of Combinatorics*, chapter 32, pages 1749–1783. North Holland, 1995.

[10] H. E. Warren. Lower bounds for approximation by nonlinear manifolds. *Transactions of the American Mathematical Society*, 133:167–178, 1968.

[11] Nathan Srebro, Jason Rennie, and Tommi Jaakkola. Maximum margin matrix factorization. In *Advances in Neural Information Processing Systems*, volume 17, 2005.

[12] Nathan Srebro. *Learning with Matrix Factorization*. PhD thesis, Massachusetts Institute of Technology, 2004.

[13] Shai Ben-David and Michael Lindenbaum. Localization vs. identification of semi-algebraic sets. *Machine Learning*, 32(3):207–224, 1998.

[14] Paul Goldberg and Mark Jerrum. Bounding the vapnik-chervonenkis dimension of concept classes parameterized by real numbers. *Machine Learning*, 18(2-3):131–148, 1995.

[15] Jacob Goodman and Richard Pollack. Upper bounds for configurations and polytopes in $\mathbb{R}^d$. *Discrete and Computational Geometry*, 1:219–227, 1986.

[16] Noga Alon. The number of polytopes, configurations and real matroids. *Mathematika*, 33:62–71, 1986.

[17] Martin Anthony and Peter L. Bartlett. *Neural Networks Learning: Theoretical Foundations*. Cambridge University Press, 1999.

[18] David Haussler. Sphere packing numbers for subsets of the boolean $n$-cube with bounded Vapnick-Chernovenkis dimension. *J. Comb. Thoery, Ser. A*, 69(2):217–232, 1995.
